# Slice sampling normalized kernel-weighted completely random measure mixture models

**Nicholas J. Foti**
Department of Computer Science
Dartmouth College
Hanover, NH 03755
nfoti@cs.dartmouth.edu

**Sinead A. Williamson**
Department of Machine Learning
Carnegie Mellon University
Pittsburgh, PA 15213
sinead@cs.cmu.edu

## Abstract

A number of dependent nonparametric processes have been proposed to model non-stationary data with unknown latent dimensionality. However, the inference algorithms are often slow and unwieldy, and are in general highly specific to a given model formulation. In this paper, we describe a large class of dependent nonparametric processes, including several existing models, and present a slice sampler that allows efficient inference across this class of models.

## 1 Introduction

Nonparametric mixture models allow us to bypass the issue of model selection, by modeling data using a random number of mixture components that can grow if we observe more data. However, such models work on the assumption that data can be considered exchangeable. This assumption often does not hold in practice as distributions commonly vary with some covariate. For example, the proportions of different species may vary across geographic regions, and the distribution over topics discussed on Twitter is likely to evolve over time.

Recently, there has been increasing interest in *dependent* nonparametric processes [1], that extend existing nonparametric distributions to non-stationary data. While a nonparametric process is a distribution over a single measure, a dependent nonparametric process is a distribution over a *collection* of measures, which may be associated with values in a covariate space. The key property of a dependent nonparametric process is that the measure at each covariate value is marginally distributed according to a known nonparametric process.

A number of dependent nonparametric processes have been developed in the literature ([2] §6). For example, the single-p DDP [1] defines a collection of Dirichlet processes with common atom sizes but variable atom locations. The order-based DDP [3] constructs a collection of Dirichlet processes using a common set of beta random variables, but permuting the order in which they are used in a stick-breaking construction. The Spatial Normalized Gamma Process (SNGP) [4] defines a gamma process on an augmented space, such that at each covariate location a subset of the atoms are available. This creates a dependent gamma process, that can be normalized to obtain a dependent Dirichlet process. The kernel beta process (KBP) [5] defines a beta process on an augmented space, and at each covariate location modulates the atom sizes using a collection of kernels, to create a collection of dependent beta processes.

Unfortunately, while such models have a number of appealing properties, inference can be challenging. While there are many similarities between existing dependent nonparametric processes, most of the inference schemes that have been proposed are highly specific, and cannot be generally applied without significant modification.

The contributions of this paper are twofold. First, in Section 2 we describe a general class of dependent nonparametric processes, based on defining completely random measures on an extended space. This class of models includes the SNGP and the KBP as special cases. Second, we develop a slice sampler that is applicable for all the dependent probability measures in this framework. We compare our slice sampler to existing inference algorithms, and show that we are able to achieve superior performance over existing algorithms. Further, the generality of our algorithm mean we are able to easily modify the assumptions of existing models to better fit the data, without the need to significantly modify our sampler.

## 2 Constructing dependent nonparametric models using kernels

In this section, we describe a general class of dependent completely random measures, that includes the kernel beta process as a special case. We then describe the class of dependent normalized random measures obtained by normalizing these dependent completely random measures, and show that the SNGP lies in this framework.

### 2.1 Kernel CRMs

A completely random measure (CRM) [6, 7] is a distribution over discrete[1] measures $B$ on some measurable space $\Omega$ such that, for any disjoint subsets $A_k \subset \Omega$, the masses $B(A_k)$ are independent. Commonly used examples of CRMs include the gamma process, the generalized gamma process, the beta process, and the stable process. A CRM is uniquely categorized by a Lévy measure $\nu(d\omega, d\pi)$ on $\Omega \times \mathbb{R}_+$, which controls the location and size of the jumps. We can interpret a CRM as a Poisson process on $\Omega \times \mathbb{R}_+$ with mean measure $\nu(d\omega, d\pi)$.

Let $\Omega = (\mathcal{X} \times \Theta)$, and let $\Pi = \{(\mu_k, \theta_k, \pi_k)\}_{k=1}^{\infty}$ be a Poisson process on the space $\mathcal{X} \times \Theta \times \mathbb{R}^+$ with associated product $\sigma$-algebra. The space has three components: $\mathcal{X}$, a bounded space of covariates; $\Theta$, a space of parameter values; and $\mathbb{R}^+$, the space of atom masses. Let the mean measure of $\Pi$ be described by the positive Lévy measure $\nu(d\mu, d\theta, d\pi)$. While the construction herein applies for any such Lévy measure, we focus on the class of Lévy measures that factorize as $\nu(d\mu, d\theta, d\pi) = R_0(d\mu)H_0(d\theta)\nu_0(d\pi)$. This corresponds to the class of homogeneous CRMs, where the size of an atom is independent of its location in $\Theta \times \mathcal{X}$, and covers most CRMs encountered in the literature. We assume that $\mathcal{X}$ is a discrete space with $P$ unique values, $\mu_p^*$, in order to simplify the exposition, and without loss of generality we assume that $R_0(\mathcal{X}) = 1$. Additionally, let $K(\cdot, \cdot) : \mathcal{X} \times \mathcal{X} \to [0, 1]$ be a bounded kernel function. Though any such kernel may be used, for concreteness we only consider a box kernel and square exponential kernel defined as

- **Box kernel:** $K(x, \mu) = \mathbf{1}\left(||x - \mu|| < W\right)$, where we call $W$ the width.

- **Square exponential kernel:** $K(x, \mu) = \exp\left(-\psi ||x - \mu||^2\right)$, for $||\cdot||$ a dissimilarity measure, and $\psi > 0$ a fixed constant.

Using the setup above we define a kernel-weighted CRM (KCRM) at a fixed covariate $x \in \mathcal{X}$ and for $A$ measurable as

$$B_x(A) = \sum_{m=1}^{\infty} K(x, \mu_m)\pi_m \delta_{\theta_m}(A) \tag{1}$$

which is seen to be a CRM on $\Theta$ by the mapping theorem for Poisson processes [8]. For a fixed set of observations $(x_1, \ldots, x_G)^T$ we define $\mathcal{B}(A) = (B_{x_1}(A), \ldots, B_{x_G}(A))^T$ as the vector of measures of the KCRM at the observed covariates. CRMs are characterized by their characteristic function (CF) [9] which for the CRM $\mathcal{B}$ can be written as

$$\mathbb{E}[\exp(-v^T \mathcal{B}(A))] = \exp\left(-\int_{\mathcal{X} \times A \times \mathbb{R}^+} (1 - \exp(-v^T \mathcal{K}_\mu \pi)\nu(d\mu, d\theta, d\pi))\right) \tag{2}$$

where $v \in \mathbb{R}^G$ and $\mathcal{K}_\mu = (K(x_1, \mu), \ldots, K(x_G, \mu))^T$. Equation 2 is easily derived from the general form of the CF of a Poisson process [8] and by noting that the one-dimensional CFs are exactly those of the individual $B_{x_i}(A)$. See [5] for a discussion of the dependence structure between $B_x$ and $B_{x'}$ for $x, x' \in \mathcal{X}$.

Taking $\nu_0$ to be the Lévy measure of a beta process [10] results in the KBP. Alternatively, taking $\nu_0$ as the Lévy measure of a gamma process, $\nu_{\mathrm{GaP}}$ [11], and $K(\cdot, \cdot)$ as the box kernel we recover the unnormalized form of the SNGP.

## 2.2 Kernel NRMs

A distribution over probability measures can be obtained by starting from a CRM, and normalizing the resulting random measure. Such distributions are often referred to as normalized random measures (NRM) [12]. The most commonly used example of an NRM is the Dirichlet process, which can be obtained as a normalized gamma process [11]. Other CRMs yield NRMs with different properties – for example a normalized generalized gamma process can have heavier tails than a Dirichlet process [13].

We can define a class of dependent NRMs in a similar manner, starting from the KCRM defined above. Since each marginal measure $B_x$ of $\mathcal{B}$ is a CRM, we can normalize it by its total mass, $B_x(\Theta)$, to produce a NRM

$$P_x(A) = B_x(A)/B_x(\Theta) = \sum_{m=1}^{\infty} \frac{K(x, \mu_m)\pi_m}{\sum_{l=1}^{\infty} K(x, \mu_l)\pi_l} \delta_{\theta_m}(A) \tag{3}$$

This formulation of a kernel NRM (KNRM) is similar to that in [14] for Ornstein-Uhlenbeck NRMs (OUNRM). While the OUNRM framework allows for arbitrary CRMs, in theory, extending it to arbitrary kernel functions is non-trivial. A fundamental difference between OUNRMs and normalized KCRMs is that the marginals of an OUNRM follow a specified process, whereas the marginals of a KCRM may be different than the underlying CRM.

A common use in statistics and machine learning for NRMs is as prior distributions for mixture models with an unbounded number of components [15]. Analogously, covariate-dependent NRMs can be used as priors for mixture models where the probability of being associated with a mixture component varies with the covariate [4, 14]. For concreteness, we limit ourselves to a kernel gamma process (KGaP) which we denote as $\mathcal{B} \sim \mathrm{KGaP}(K, R_0, H_0, \nu_{\mathrm{GaP}})$, although the slice sampler can be adapted to any normalized KCRM.

Specifically, we observe data $\{(x_j, y_j)\}_{j=1}^N$ where $x_j \in \mathcal{X}$ denotes the covariate of observation $j$ and $y_j \in \mathbb{R}^d$ denotes the quantities we wish to model. Let $x_g^*$ denote the $g$th unique covariate value among all the $x_j$ which induces a partition on the observations so that observation $j$ belongs to group $g$ if $x_j = x_g^*$. We denote the $i$th observation corresponding to $x_g^*$ as $y_{g,i}$.

Each observation is associated with a mixture component which we denote as $s_{g,i}$ which is drawn according to a normalized KGaP on a parameter space $\Theta$, such that $(\theta, \phi) \in \Theta$, where $\theta$ is a mean and $\phi$ a precision. Conditional on $s_{g,i}$, each observation is then drawn from some density $q(\cdot|\theta, \phi)$ which we assume to be $N(\theta, \phi^{-1})$. The full model can then be specified as

$$P_g(A)|\mathcal{B} \sim B_g(A)/B_g(\Theta)$$
$$s_{g,i}|P_g \sim \sum_{m=1}^{\infty} \frac{K(x_g^*, \mu_m)\pi_m}{\sum_{l=1}^{\infty} K(x_g^*, \mu_l)\pi_l} \delta_m \tag{4}$$
$$(\theta_m^*, \phi_m^*) \sim H_0(d\theta, d\phi)$$
$$y_{g,i}|s_{g,i}, \{(\theta^*, \phi^*)\} \sim q(y_{g,i}|\theta_{s_{g,i}}^*, \phi_{s_{g,i}}^*)$$

If $K(\cdot, \cdot)$ is a box kernel, Eq. 4 describes a SNGP mixture model [4].

# 3 A slice sampler for dependent NRMs

The slice sampler of [16] allows us to perform inference in arbitrary NRMs. We extend this slice sampler to perform inference in the class of dependent NRMs described in Sec. 2.2. The slice sampler can be used with any underlying CRM, but for simplicity we concentrate on an underlying gamma process, as described in Eq. 4. In the supplement we also derive a Rao-Blackwellized estimator of the predictive density for unobserved data using the output from the slice sampler. We use this estimator to compute predictive densities in the experiments.

Analogously to [16] we introduce a set of auxiliary slice variables – one for each data point. Each data point can only belong to clusters corresponding to atoms larger than its slice variable. The set of slice variables thus defines a minimum atom size that need be represented, ensuring a finite number of instantiated atoms.

We extend this idea to the KNRM framework. Note that, in this case, an atom will exhibit different sizes at different covariate locations. We refer to these sizes as the *kernelized atom sizes*, $K(x_g^*, \mu)\pi$, obtained by applying a kernel $K$, evaluated at location $x_g^*$, to the raw atom $\pi$. Following [16], we introduce a local slice variable $u_{g,i}$. This allows us to write the joint distribution over the data points $y_{g,i}$, their cluster allocations $s_{g,i}$ and their slice variables $u_{g,i}$ as

$$f(\mathbf{y}, \mathbf{u}, \mathbf{s} | \pi, \mu, \theta, \phi) = \prod_{g=1}^{G} V_g^{n_g-1} e^{(-V_g B_{Tg})} \prod_{i=1}^{n_g} \mathbf{1} \left( u_{g,i} < K(x_g^*, \mu_{s_{g,i}})\pi_{s_{g,i}} \right) q(y_{g,i} | \theta_{s_{g,i}}, \phi_{s_{g,i}})$$
(5)

where $B_{Tg} = B_{x_g^*}(\Theta) = \sum_{m=1}^{\infty} K(x_g^*, \mu_m)\pi_m$ and $V_g \sim \text{Ga}(n_g, B_{Tg})$ is an auxiliary variable[2]. See the supplement and [16, 17] for a complete derivation.

In order to evaluate Eq. 5, we need to evaluate $B_{Tg}$, the total mass of the unnormalized CRM at each covariate value. This involves summing over an infinite number of atoms – which we do not wish to represent. Define $0 < L = \min \{u_{s_{g,i}}\}$. This gives the smallest possible (kernelized) atom size to which data can be attached. Therefore, if we instantiate all atoms with raw size greater than $L$, we will include all atoms associated with occupied clusters. For any value of $L$, there will be a finite number $M$ of atoms above this threshold. From these $M$ raw atoms, we can obtain the kernelized atoms above the slice corresponding to a given data point.

We must obtain the remaining mass by marginalizing over all kernelized atoms that are below the slice (see the supplement). We can split this mass into, **a**.) the mass due to atoms that are not instantiated (i.e. whose kernelized value is below the slice at all covariate locations) and, **b**.) the mass due to currently instantiated atoms (i.e. atoms whose kernelized value is above the slice at at least one covariate location) [3]. As we show in the supplement, the first term, **a**, corresponds to atoms $(\pi, \mu)$ where $\pi < L$, the mass of which can be written as

$$\sum_{\mu^* \in \mathcal{X}} \left( R_0(\mu^*) \int_0^L (1 - \exp(-V^T \mathcal{K}_{\mu^*} \pi)) \nu_0(d\pi) \right)$$
(6)

where $V = (V_1, \ldots, V_G)^T$. This can be evaluated numerically for many CRMs including gamma and generalized gamma processes [16]. The second term, **b**, consists of realized atoms $\{(\pi_k, \mu_k)\}$ such that $K(x_g^*, \mu_k)\pi_k < L$ at covariate $x_g^*$. We use a Monte Carlo estimate for **b** that we describe in the supplement. For box kernels term **b** vanishes, and we have found that even for the square exponential kernel ignoring this term yields good results.

### 3.1 Sampling equations

Having specified the joint distribution in terms of a finite measure with a random truncation point $L$ we can now describe a sampler that samples in turn from the conditional distributions for the auxiliary variables $V_g$, the gamma process parameter $\alpha = H_0(\Theta)$, the instantiated raw atom sizes $\pi_m$ and corresponding locations in covariate space $\mu_m$ and in parameter space $(\theta_m, \phi_m)$, and the slice variables $u_{g,i}$. We define some simplifying notation: $\mathcal{K}_\mu = (K(x_1^*, \mu), \ldots, K(x_G^*, \mu))^T$; $B_+ = (B_{+1}, \ldots, B_{+G})^T$, $B_* = (B_{*1}, \ldots, B_{*G})^T$, where $B_{+g} = \sum_{m=1}^{M} K(x_g^*, \mu_m)\pi_m$, $B_{*g} = \sum_{m=M+1}^{\infty} K(x_g^*, \mu_m)\pi_m$ so that $B_{Tg} = B_{+g} + B_{*g}$; and $n_{g,m} = |\{s_{g,i} : s_{g,i} = m, i \in 1, \ldots, n_g\}|$.

- **Auxiliary variables $V_g$**: The full conditional distribution for $V_g$ is given by

$$p(V_g \mid n_g, V_{-g}, B_+, B_*) \propto V_g^{n_g-1} \exp(-V^T B_+) \mathbb{E}[\exp(-V^T B_*)], \ V_g > 0 \quad (7)$$

which we sample using Metropolis-Hastings moves, as in [18].

- **Gamma process parameter** $\alpha$: The conditional distribution for $\alpha$ is given by

$$p(\alpha \mid K, V, \mu, \pi) \propto p(\alpha)\alpha^K e^{-\alpha\left[\int_L^\infty \nu_0(d\pi) + \int_0^L \int_{\mathcal{X}} (1-\exp(-V^T \mathcal{K}_\mu \pi))R_0(d\mu)\nu_0(d\pi)\right]} \quad (8)$$

  If $p(\alpha) = \mathrm{Ga}(a_0, b_0)$ then the posterior is also a gamma distribution with parameters

$$a = a_0 + K \quad (9)$$

$$b = b_0 + \int_L^\infty \nu_0(d\pi) + \int_{\mathcal{X}} \int_0^L (1 - \exp(-V^T \mathcal{K}_\mu \pi))\nu_0(d\pi)R_0(d\mu) \quad (10)$$

  where the first integral in Eq. 10 can be evaluated for many processes of interest and the second integral can be evaluated as in Eq. 6.

- **Raw atom sizes** $\pi_m$: The posterior for atoms associated with occupied clusters is given by

$$p(\pi_m \mid n_{g,m}, \mu_m, V, B_+) \propto \pi_m^{\sum_{g=1}^G n_{g,m}} \exp\left(-\pi_m \sum_{g=1}^G V_g K(x_g^*, \mu_m)\right) \nu_0(\pi_m) \quad (11)$$

  For an underlying gamma or generalized gamma process, the posterior of $\pi_m$ will be given by a gamma distribution due to conjugacy [16]. There will also be a number of atoms with raw size $\pi_m > L$ that do not have associated data. The number of such atoms is Poisson distributed with mean $\alpha \int_A \exp(-V^T \mathcal{K}_\mu \pi)\nu_0(d\pi)R_0(d\pi)$, where $A = \{(\mu, \pi) : K(x_g^*, \mu)\pi > L, \text{ for some } g\}$ and which can be computed using the approach described for Eq. 6.

- **Raw atom covariate locations** $\mu_m$: Since we assume a finite set of covariate locations, we can sample $\mu_m$ according to the discrete distribution

$$p(\mu_m \mid n_{g,m}, V, B_+) \propto \prod_{g=1}^G K(x_g^*, \mu_k)^{n_{g,m}} \exp\left(-\pi_m \sum_{g=1}^G V_g K(x_g^*, \mu_m)\right) R_0(\mu_m) \quad (12)$$

- **Slice variables** $u_{g,i}$: Sampled as $u_{g,i} \mid \{\pi\}, \{\mu\}, s_{g,i} \sim \mathrm{Un}[0, K(x_g^*, \mu_{s_{g,i}})\pi_{s_{g,i}}]$.

- **Cluster allocations** $s_{g,i}$: The prior on $s_{g,i}$ cancels with the prior on $u_{g,i}$, yielding

$$p(s_{g,i} = m \mid y_{g,i}, u_{g,i}, \theta_m, \pi_m, \mu_m) \propto q(y_{g,i}|\theta_m, \phi_m)\mathbf{1}\left(u_{g,i} < K(x_g^*, \mu_m)\pi_m\right) \quad (13)$$

  where only a finite number of $m$ need be evaluated.

- **Parameter locations**: Can be sampled as in a standard mixture model [16].

## 4  Experiments

We evaluate the performance of the proposed slice sampler in the setting of covariate dependent density estimation. We assume the statistical model in Eq. 4 and consider a univariate Gaussian distribution as the data generating distribution. We use both synthetic and real data sets in our experiments and compare the slice sampler to a Gibbs sampler for a finite approximation to the model (see the supplement for details of the model and sampler) and to the original SNGP sampler. We assess the mixing characteristics of the sampler using the integrated autocorrelation time $\tau$ of the number of clusters used by the sampler at each iteration after a burn-in period, and by the predictive quality of the collective samples on held-out data. The integrated autocorrelation time of samples drawn from an MCMC algorithm controls the Monte Carlo error inherent in a sample drawn from the MCMC algorithm. It can be shown that in a set of $T$ samples from the MCMC algorithm, there are in effect only $T/(2\tau)$ "independent" samples. Therefore, lower values of $\tau$ are deemed better. We obtain an estimate $\hat{\tau}$ of the integrated autocorrelation time following [19].

We assess the predictive performance of the collected samples from the various algorithms by computing a Monte Carlo estimate of the predictive log-likelihood of a held-out data point under the model. Specifically, for a held out point $y^*$ we have

$$\log p(y^*|y) \approx \frac{1}{T}\sum_{t=1}^T \log\left(\sum_{m=1}^{M^{(t)}} w_m^{(t)} q\left(y^*|\theta_m^{(t)}, \phi_m^{(t)}\right)\right). \quad (14)$$

Table 1: Results of the samplers using different kernels. Entries are of the form "average predictive density / average number of clusters used / $\hat{\tau}$" where two standard errors are shown in parentheses. Results are averaged over 5 hold-out data sets.

|  | **Synthetic** | **CMB** | **Motorcycle** |
|---|---|---|---|
| Slice Box | -2.70 (0.12) / 11.6 / 2442 | -0.15 (0.11) / 14.4 / 2465 | -0.90 (0.28) / 10.3 / 2414 |
| SNGP | -2.67 (0.12) / 43.3 / 2488 | -0.22 (0.14) / 79.1 / 2495 | NA |
| Finite Box | -2.78 (0.15) / 11.7 / 2497 | -0.41 (0.14) / 18.2 / 2444 | -1.19 (0.16) / 16.4 / 2352 |
| Slice SE | NA | -0.28 (0.07) / 14.7 / 2447 | -0.87 (0.28) / 8.2 / 2377 |
| Finite SE | NA | -0.29 (0.05) / 9.5 / 2491 | -0.99 (0.19) / 7.3 / 2159 |

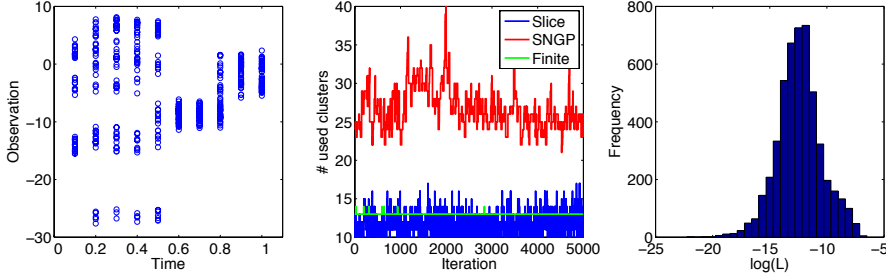

Figure 1: Left: Synthetic data. Middle: Trace plots of the number of clusters used by the three samplers. Right: Histogram of truncation point $L$.

The weight $w_m^{(t)}$ is the probability of choosing atom $m$ for sample $t$. We did not use the Rao-Blackwellized estimator to compute Eq. 14 for the slice sampler to achieve fair comparisons (see the supplement for the results using the Rao-Blackwellized estimator).

## 4.1 Synthetic data

We generated synthetic data from a dynamic mixture model with 12 components (Figure 1). Each component has an associated location, $\mu_k$, that can take the value of any of ten uniformly spaced time stamps, $t_j \in [0, 1]$. The components are active according to the kernel $K(x, \mu_k) = \mathbf{1}\left(|x - \mu_k| < .2\right)$ – i.e. components are active for two time stamps around their location. At each time stamp, $t_j$, we generate 60 data points. For each data point we choose a component, $k$, such that $\mathbf{1}\left(|t_j - \mu_k| < .2\right)$ and then generate that data point from a Gaussian distribution with mean $\mu_k$ and variance 10. We use 50 of the generated data points per time stamp as a training set and hold out 10 data points for prediction.

Since the SNGP is a special case of the normalized KGaP, we compare the finite and slice samplers, which are both conditional samplers, to the original marginal sampler proposed in [4]. We use the basic version of the SNGP that uses fixed-width kernels, as we assume fixed width kernel functions for simplicity. The implementation of the SNGP sampler we used also only allows for fixed component variances, so we fix all $\phi_k = 1/10$, the true data generating precision. We use the true kernel function that was used to generate the data as the kernel for the normalized KGaP model.

We ran the slice sampler for $10,000$ burn-in iterations and subsequently collected $5,000$ samples. We truncated the finite version of the model to 100 atoms and ran the sampler for $5,000$ burn-in iterations and collected $5,000$ samples. The SNGP sampler was run for $2,000$ burn-in iterations and $5,000$ samples were collected[4]. The predictive log-likelihood, mean number of clusters used and $\hat{\tau}$ are shown in the "Synthetic" column in Table 1.

We see that all three algorithms find a region of the posterior that gives predictive estimates of a similar quality. The autocorrelation estimates for the three samplers are also very similar. This might seem surprising, since the SNGP sampler uses sophisticated split-merge moves to improve mixing, which have no analogue in the slice sampler. In addition, we note that although the per-iteration

mixing performance is comparable, the average time per 100 iterations for the slice sampler was $\sim 10$ seconds, for the SNGP sampler was $\sim 30$ seconds and for the finite sampler was $\sim 200$ seconds. Even with only 100 atoms the finite sampler is much more expensive than the slice and SNGP[5] samplers.

We also observe (Figure 1) that both the slice and finite samplers use essentially the true number of components underlying the data and that the SNGP sampler uses on average twice as many components. The finite sampler finds a posterior mode with 13 clusters and rarely makes small moves from that mode. The slice sampler explores modes with 10-17 clusters, but never makes large jumps away from this region. The SNGP sampler explores the largest number of used clusters ranging from 23-40, however, it has not explored regions that use less clusters.

Figure 1 also depicts the distribution of the variable truncation level $L$ over all samples in the slice sampler. This suggests that a finite model that discards atoms with $\pi_k < 10^{-18}$ introduces negligible truncation error. However, this value of $L$ corresponds to $\approx 10^{18}$ atoms in the finite model which is computationally intractable. To keep the computation times reasonable we were only able to use 100 atoms, a far cry from the number implied by $L$.

In Figure 2 (Left) we plot estimates of the predictive density at each time stamp for the slice (a), finite (b) and SNGP (c) samplers. All three samplers capture the evolving structure of the distribution. However, the finite sampler seems unable to discard unneeded components. This is evidenced by the small mass of probability that spans times $[0, 0.8]$ when the data that the component explains only exists at times $[0.2, 0.5]$. The slice and SNGP samplers seem to both provide reasonable explanations for the distribution, with the slice sampler tending to provide smoother estimates.

## 4.2 Real data

As well as providing an alternative inference method for existing models, our slice sampler can be used in a range of models that fall under the general class of KNRMs. To demonstrate this, we use the finite and slice versions of our sampler to learn two kernel DPs, one using a box kernel, $K(x, \mu) = \mathbf{1}\,(|x - \mu| < 0.2)$ (the setting in the SNGP), and the other using a square exponential kernel $K(x, \mu) = \exp(-200(x - \mu)^2)$, which has support approximately on $[\mu - .2, \mu + .2]$. The kernel was chosen to be somewhat comparable to the box kernel, however, this kernel allows the influence of an atom to diminish gradually as opposed to being constant. We compare to the SNGP sampler for the box kernel model, but note that this sampler is not applicable to the exponential kernel model.

We compare these approaches on two real-world datasets:

- **Cosmic microwave background radiation (CMB)**[20]: TT power spectrum measurements, $\eta$, from the cosmic microwave background radiation (CMB) at various 'multipole moments', denoted $M$. Both variables are considered continuous and exhibit dependence. We rescale $M$ to be in $[0, 1]$ and standardize $\eta$ to have mean 0 and unit variance.

- **Motorcycle crash data** [21]. This data set records the head acceleration, $A$, at various times during a simulated motorcycle crash. We normalize time to $[0, 1]$ and standardize $A$ to have mean 0 and unit variance.

Both datasets exhibit local heteroskedasticity, which cannot be captured using the SNGP. For the CMB data, we consider only the first 600 multipole moments, where the variance is approximately constant, allowing us to compare the SNGP sampler to the other algorithms. For all models we fixed the observation variance to 0.02, which we estimated from the standardized data. To ease the computational burden of the samplers we picked 18 time stamps in $[0.05, 0.95]$, equally spaced 0.05 apart and assigned each observation to the time stamp closest to its associated value of $M$. This step is by no means necessary, but the running time of the algorithms improves significantly. For the

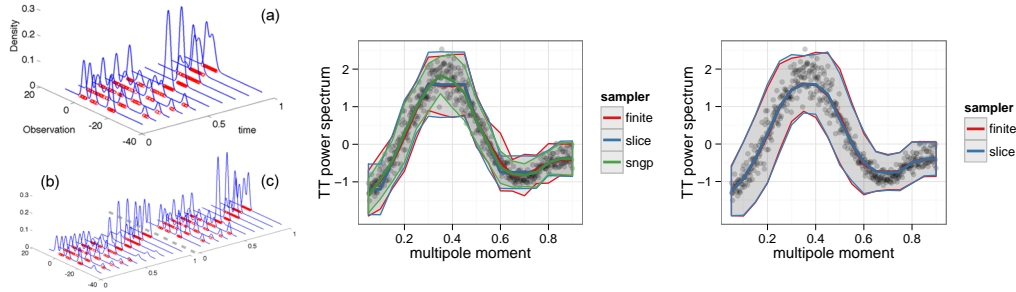

Figure 2: **Left:** Predictive density at each time stamp for synthetic data using the slice (a), finite (b) and SNGP (c) samplers. The scales of all three axis are identical. **Middle:** Mean and 95% CI of predictive distribution for all three samplers on CMB data using the box kernel. **Right:** Mean and 95% CI of predictive distribution using the square exponential kernel.

motorcycle data, there was no regime of constant variance, so we only compare the slice and finite truncation samplers[6].

For each dataset and each model/sampler, the held-out predictive log-likelihood, the mean number of used clusters and $\hat{\tau}$ are reported in Table 1. The mixing characteristics of the chain are similar to those obtained for the synthetic data. We see in Table 1 that the box kernel and the square exponential kernel produce similar results on the CMB data. However, the kernel width was not optimized and different values may prove to yield superior results. For the motorcycle data we see a noticeable difference between using the box and square exponential kernels where using the latter improves the held-out predictive likelihood and results in both samplers using fewer components on average.

Figure 2 shows the predictive distributions obtained on the CMB data. Looking at the mean and 95% CI of the predictive distribution (middle) we see that when using the box kernel the SNGP actually fits the data the best. This is most likely due to the fact that the SNGP is using more atoms than the slice or finite samplers. We show that the square exponential kernel (right) gives much smoother estimates and appears to fit the data better, using the same number of atoms as were learned with the box kernel (see Table 1). We note that the slice sampler took $\sim 20$ seconds per 100 iterations while the finite sampler used $\sim 150$ seconds.

## 5 Conclusion

We presented the class of normalized kernel CRMs, a type of dependent normalized random measure. This class generalizes previous work by allowing more flexibility in the underlying CRM and kernel function used to induce dependence. We developed a slice sampler to perform inference on the infinite dimensional measure and compared this method with samplers for a finite approximation and for the SNGP. We found that the slice sampler yields samples with competitive predictive accuracy at a fraction of the computational cost.

There are many directions for future research. Incorporating reversible-jump moves [22] such as split-merge proposals should allow the slice sampler to explore larger regions of the parameter space with a limited decrease in computational efficiency. A similar methodology may yield efficient inference algorithms for KCRMs such as the KBP, extending the existing slice sampler for the Indian Buffet Process [23].

**Acknowledgments**

NF was funded by grant AFOSR FA9550-11-1-0166. SW was funded by grants NIH R01GM087694 and AFOSR FA9550010247.

## Footnotes

[1] with, possibly, a deterministic continuous component

[2]We parametrize the gamma distribution so that $X \sim \text{Ga}(a, b)$ has mean $a/b$ and variance $a/b^2$

[3]If $\mathcal{X}$ were not bounded there would be a third term consisting of raw atoms $> L$ that when kernelized fall below the slice everywhere. These can be ignored by a judicious choice of the space $\mathcal{X}$ and the allowable kernel widths.

[4]No thinning was performed in any of the experiments in this paper.

[5]Sampling the cluster means and assignments is the slowest step for the SNGP sampler taking about 3 seconds. The times reported here only performed this step every 25 iterations achieving reasonable results. If this step were performed every iteration the results may improve, but the computation time will explode.

[6]The SNGP could still be used to model this data, however, then we would be comparing the models as opposed to the samplers.

# References

[1] S.N. MacEachern. Dependent nonparametric processes. In *ASA Proceedings of the Section on Bayesian Statistical Science*, 1999.

[2] D. Dunson. Nonparametric Bayes applications to biostatistics. In N. L. Hjort, C. Holmes, P. Müller, and S. G. Walker, editors, *Bayesian Nonparametrics*. Cambridge University Press, 2010.

[3] J.E. Griffin and M.F.J. Steel. Order-based dependent Dirichlet processes. *JASA*, 101(473):179–194, 2006.

[4] V. Rao and Y.W. Teh. Spatial normalized gamma processes. In *NIPS*, 2009.

[5] L. Ren, Y. Wang, D. Dunson, and L. Carin. The kernel beta process. In *NIPS*, 2011.

[6] J.F.C. Kingman. Completely random measures. *Pacific Journal of Mathematics*, 21(1):59–78, 1967.

[7] A. Lijoi and I. Prünster. Models beyond the Dirichlet process. Technical Report 129, Collegio Carlo Alberto, 2009.

[8] J.F.C. Kingman. *Poisson processes*. OUP, 1993.

[9] B. Fristedt and L.F. Gray. *A Modern Approach to Probability Theory*. Probability and Its Applications. Birkhäuser, 1997.

[10] N.L. Hjort. Nonparametric Bayes estimators based on beta processes in models for life history data. *Annals of Statistics*, 18:1259–1294, 1990.

[11] T.S. Ferguson. A Bayesian analysis of some nonparametric problems. *Annals of Statistics*, 1(2):209–230, 1973.

[12] E. Regazzini, A. Lijoi, and I. Prünster. Distributional results for means of normalized random measures with independent increments. *Annals of Statistics*, 31(2):pp. 560–585, 2003.

[13] A. Lijoi, R.H. Mena, and I. Prünster. Controlling the reinforcement in Bayesian non-parametric mixture models. *JRSS B*, 69(4):715–740, 2007.

[14] J.E. Griffin. The Ornstein-Uhlenbeck Dirichlet process and other time-varying processes for Bayesian nonparametric inference. Technical report, Department of Statistics, University of Warwick, 2007.

[15] S. Favaro and Y.W. Teh. MCMC for normalized random measure mixture models. *Submitted*, 2012.

[16] J. E. Griffin and S. G. Walker. Posterior simulation of normalized random measure mixtures. *Journal of Computational and Graphical Statistics*, 20(1):241–259, 2011.

[17] L.F. James, A. Lijoi, and I. Prünster. Posterior analysis for normalized random measures with independent increments. *Scandinavian Journal of Statistics*, 36(1):76–97, 2009.

[18] J.E. Griffin, M. Kolossiatis, and M.F.J. Steel. Comparing distributions using dependent normalized random measure mixtures. Technical report, University of Warwick, 2010.

[19] M. Kalli, J.E. Griffin, and S.G. Walker. Slice sampling mixture models. *Statistics and Computing*, 21(1):93–105, 2011.

[20] C.L. Bennett et al. First year Wilkinson Microwave Anisotropy Probe (WMAP) observations: Preliminary maps and basic results. *Astrophysics Journal Supplement*, 148:1, 2003.

[21] B.W. Silverman. Some aspects of the spline smoothing approach to non-parametric curve fitting. *JRSS B*, 47:1–52, 1985.

[22] P.J. Green. Reversible jump Markov chain Monte Carlo computation and Bayesian model determination. *Biometrika*, 82(4):711–732, 1995.

[23] Y.W. Teh, D. Görür, and Z. Ghahramani. Stick-breaking construction for the Indian buffet process. In *AISTATS*, volume 11, 2007.

